# Multiple timescales and uncertainty in motor adaptation

**Konrad P. Körding**
Rehabilitation Institute of Chicago
Northwestern University, Dept. PM&R
Chicago, IL 60611
`konrad@koerding.com`

**Joshua B. Tenenbaum**
Massachusetts Institute of Technology
Cambridge, MA 02139
`jbt@mit.edu`

**Reza Shadmehr**
Johns Hopkins University
Baltimore, MD 21205
`reza@bme.jhu.edu`

## Abstract

Our motor system changes due to causes that span multiple timescales. For example, muscle response can change because of fatigue, a condition where the disturbance has a fast timescale or because of disease where the disturbance is much slower. Here we hypothesize that the nervous system adapts in a way that reflects the temporal properties of such potential disturbances. According to a Bayesian formulation of this idea, movement error results in a credit assignment problem: what timescale is responsible for this disturbance? The adaptation schedule influences the behavior of the optimal learner, changing estimates at different timescales as well as the uncertainty. A system that adapts in this way predicts many properties observed in saccadic gain adaptation. It well predicts the timecourses of motor adaptation in cases of partial sensory deprivation and reversals of the adaptation direction.

## 1 Introduction

Saccades are rapid eye movements that shift the direction of gaze from one target to another. The eyes move so fast [1] that visual feedback can not usually be used during the movement. For that reason, without adaptation any changes in the properties of the oculomotor plant would lead to inaccurate saccades [2]. Motor gain is the ratio of actual and desired movement distances. If the motor gain decreases to below one then the nervous system must send a stronger command to produce a movement of the same size. Indeed, it has been observed that if saccades overshoot the target, the gain tends to decrease and if they undershoot, the gain tends to increase. The saccadic jump paradigm [3] is often used to probe such adaptation [4]: while the subject moves its eyes towards a target, the target is moved. This is not distinguishable to the subject from a change in the properties of the oculomotor plant [5]. Using this paradigm it is possible to probe the mechanism that is normally used to adapt to ongoing changes of the oculomotor plant.

### 1.1 Disturbances to the motor plant

Properties of the oculomotor plant may change due to a variety of disturbances, such as various kinds of fatigue and disease. The fundamental characteristic of these disturbances is that their effects unfold over a wide range of timescales. Here we model each disturbance as a random walk with a

characteristic timescale (Figures 1A and B) over which the disturbance is expected to go away.

$$disturbance_\tau(t + \Delta) = (1 - 1/\tau)disturbance_\tau(t) + \epsilon_\tau \qquad (1)$$

where $\epsilon_\tau$ is drawn from a mean zero normal distribution of width $\sigma_\tau$, and $\tau$ is the timescale. The larger $\tau$ the closer $(1 - 1/\tau$ is to 1 and the longer does a disturbance typically last.

## 1.2 Parameter choice

For the experiments that we want to explain only those timescales will matter that are not much longer than the overall time of the experiments (because they would already have been integrated out) and that are not much shorter than the time of an individual saccade (because they would average out). For that reason we chose the distribution of $\tau$ to be 30 values exponentially scaled between 1 and 33333 saccades. The distribution of expected gains thus only depends on the distribution of $\sigma_\tau$, a characterization of how important disturbances are at various timescales. It seems plausible that disturbances that have a short timescale tend to be more variable than those that have a long timescale, and we choose: $\sigma_\tau = c/\tau$ where $c$ is one of the two free parameters of our model. Moreover, as we expect each disturbance to be relatively small, we assume linearity and that the motor gain is simply one plus the sum of all the disturbances:

$$gain(t) = 1 + \sum_\tau disturbance_\tau(t) \qquad (2)$$

If the motor plant underwent such changes in its properties, and if the nervous system produced the same motor commands without adaptation, then saccade gain would differ from one, resulting in motor error. However, with each saccade, the brain observes consequences of the motor commands. We assume that this observation is corrupted by noise:

$$observation(t) = gain(t) + w \qquad (3)$$

where $w$ is the second free parameter of our model, the observation noise with a width $\sigma_w$. Throughout this paper we choose $\sigma_w = 0.05$ which we estimated from the spread of saccade gains over typical periods of 200 saccades and $c = 0.002$ because that yielded good fits to the data by Hopp and Fuchs [2]. We chose to model all data using the same set of parameters to avoid issues of overfitting.

## 1.3 Inference

Given this explicit model, Bayesian statistics allows deriving an optimal adaptation strategy. We observe that the system is equivalent to the generative model of the Kalman filter [6] with a diagonal transition matrix $M = diag(1 - 1/\tau)$ and an observation matrix $H$ that is a vector consisting of one 1 for each of the 30 potential disturbances, and a diagonal process noise matrix of $Q = diag(\tau^{-1})$. Process noise is what is driving the changes of each of the disturbances. We obtain the solution that is well known from the Kalman Filter literature. We use the Kalman filter toolbox written by Kevin Murphy to numerically solve these equations.

An optimally adapting system needs to explicitly represent contribution of each timescale. Because the contribution of each timescale can never be known precisely, the Bayesian learner represents what it knows as a probability distribution. As the model is linear and the noises are Gaussian, it is sufficient to keep first and second order statistics. And so the learner represents what it knows about the contribution of each timescale as a best estimate, but also keeps a measure of uncertainty around this estimate (Fig 1C). Any point along the +0% gain line is a point where the fast and slow timescale cancel each other. There is a line associated with any possible gain (e.g. +30% and -30%). Every timestep the system starts with its belief that it has from the previous timestep (sketched in yellow) and combines this with information from the current saccade (sketched in blue) to come up with a new estimate (sketched in red). Two important changes happen to the belief of the learner over time. (1) When time passes, disturbances can be expected to get smaller but at the same time our uncertainty about them increases. (2) when a movement error is observed then this biases the sum of the disturbances to the observed error value and it also decreases the uncertainty. These effects are sketched in Figure 1D. Normally the adaptation mechanism is responding to the small drifts that happen to the oculomotor plant and the estimate from the saccade is largely overlapping with the prior belief and with the new belief. When the light is turned off the estimate of each of the

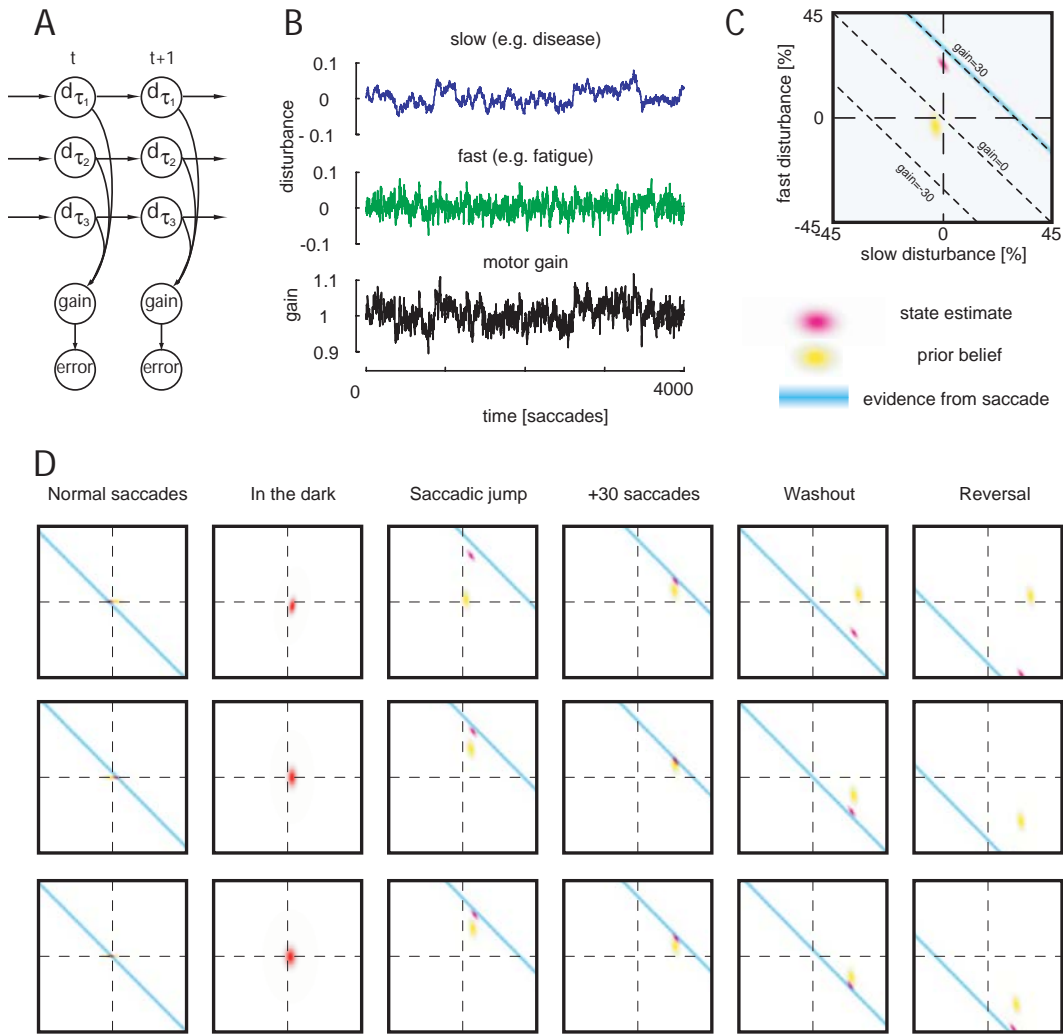

Figure 1: A generative model for changes in the motor plant and the corresponding optimal inference. A) Various disturbances $d$ evolve over time as independent random walks. The gain is a linear function of all these random walks. The observed error is a noisy version of the gain. B) An example of a system with two timescales (fast and slow), and the resulting gain. C) Optimal inference during a saccade adaptation experiment. For illustrative purposes, here we assume only two timescales. The yellow cloud represents the learners belief about the current combination of disturbances (prior). The system observes a saccade with an error of +30%. The region about the blue line is the uncertainty about the observation (i.e., the likelihood). Combining this information with the prior belief (yellow) leads to the posterior estimate (red). After a single observation of the +30% condition the most probable estimate thus is that it is a fast disturbance. D) The changes of estimates under various perturbations. Here we simulated a saccade on every 10th time step of the model. Each column shows three consecutive trials (top to bottom). Only in the darkness case saccades 1 3 and 50 are shown. In the dark, parameter uncertainties increase because the learner is not allowed to make observations (sensory noise is effectively infinite). In a gain increase paradigm, initially most of the error is associated with the fast perturbations. After 30 saccades in the gain increase paradigm, most of the error is associated with slow perturbations. Washout trials that follow gain increase do not return the system to a naive state. Rather, estimates of fast and slow perturbations cancel each other. Gain decrease following gain increase training will mostly affect the fast system.

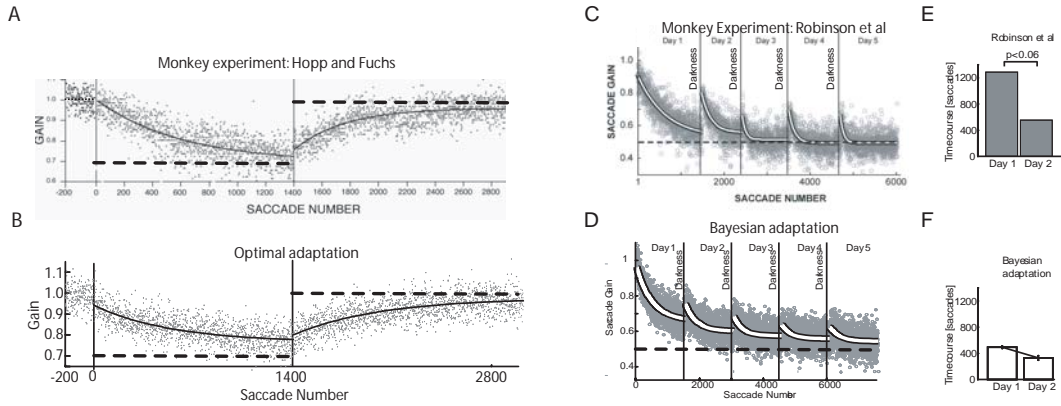

Figure 2: Saccadic gain adaptation in a target jump paradigm. A) Data replotted from Hopp and Fuchs [2] with permission. Each dot is one saccade, the thick lines are exponential fits to the intervals [0 1400] and [1400:2800]. Starting at saccade number 0 the target is jumping 30% short of the target giving the impression of muscles that are too strong. The gain then decreases until the manipulation is ended at saccade number 1400. B) The same plot is shown for the optimal Bayesian learner. Changes without feedback. C) Data reprinted from [7]. Normal saccadic gain change paradigm as in Figure 2, however now the monkey spends its nights without vision and the paradigm is continued for many days. D) The same plot as in C) but for the Bayesian learner. E) Comparison of the saccadic gain change timecourses obtained by fitting an exponential. F) the same figure as in E) for the Bayesian learner

disturbances slowly creeps towards zero. At the same time, however the uncertainty increases a lot and larger uncertainty allows faster learning because the new information is more precise than the prior information. In the saccadic jump paradigm the error is much larger than it would be during normal life and this is first interpreted by the learner as a fast change and as it persists progressively interpreted as a slow change. When the saccadic jumps ends then the fast timescale goes negative fast and the slow timescale slowly approaches zero. In a reversal setting the fast timescale becomes very negative and the slow timescale goes towards zero. Already with two timescales the optimal learner can thus exhibit a large number of interesting properties.

## 2 Results: Comparison with experimental data

### 2.1 Saccadic gain adaptation

In an impressive range of experiments started by Mclaughlin [3], investigators have examined how monkeys adapt their saccadic gain. Figure 2A shows how the gain changes over time so that saccades progressively become more precise. The rate of adaptation typically starts fast and then progressively gets slower. This is a classic pattern that is reflected in numerous motor adaptation paradigms [8, 9]. The same patterns are seen for the Bayesian multiscale learner (Figure 2B). Fast timescale disturbances are assumed to increase and decrease faster than slow timescale disturbances. Therefore, when the gain rapidly changes, it is a priori most likely that it will go away fast. (Fig. 1D, saccadic jump). Between trials, the estimates of the fast disturbances decay fast, but this decay is smaller in the slower timescales. If the gain change is maintained, the relative contribution of the fast timescales diminishes in comparison to the slow timescales (Fig. 1D, +30 saccades). As fast timescales adapt fast but decay fast as well and slow timescales adapt and decay slowly, this implies that the gain change is driven by progressively slower timescales resulting in the transition from initial fast adapting to a progressively slower adapting.

### 2.2 Saccadic gain adaptation after sensory deprivation

The effects of a wide range of timescales and uncertainty about the causes of changes of the oculo-motor plant will largely be hidden if experiments are of a relatively short duration and no uncertainty

is produced. However, in a recent experiment Robinson et al analyzed saccadic gain adaptation [7] in a way that allowed insight into many timescales as well as insight into the way the nervous system deals with uncertainty. The adaptation target was set to -50%. The monkey adapted for about 1500 saccades every day for 21 consecutive days. Because of the long duration many different timescales are involved in this process. Interestingly, during the rest of the day the monkey wore goggles that blocked vision. During these breaks monkeys will accumulate uncertainty about the state of their oculomotor plant. Figure 2C shows results from such an experiment and figure 2D shows the results we are getting from the Bayesian learner. The results are surprisingly similar given that we used the same parameters that we had used the model parameters inferred from the Hopp and Fuchs data. Two effects are visible in the data. (1) There are several timescales during adaptation: there is a fast (100 saccades) and a slow (10 days) timescale. Closer examination of the data reveals a wide spectrum of timescales. (2) The state estimate is affected by the periods of darkness. During the breaks that are paired with darkness the system is decaying back to a gain of zero, as predicted by the model. Moreover, darkness leads to increased uncertainty. Increased uncertainty means that new information is relatively more precise than old information which in turn leads to faster learning. Consequently monkeys learn faster during the second day (after spending a night without feedback) than during the first (quantified in figure 2E and F). The finding that the Bayesian learner seems to change faster than the monkey may be related to the context being somewhat different than in the Hopp and Fuchs experiment. The system seems to represent uncertainty and clearly represents the way the motor plant is expected to change in the absence of feedback. It has been proposed that the nervous system may use a set of integrators where one is learning fast and the other is learning slowly [10, 11]. The Bayesian learner, however, keeps a measure of uncertainty about its estimates. For that reason only the Bayesian learner can explain the fact that sensory deprivation appears to enhance learning rates.

## 2.3   Gain adaptation with reversals

Kojima et al [12] reported a host of surprising behavioral results during saccade adaptation. In these experiments the adaptation direction was changed 3 times. The saccadic gain was initially increased, then decreased until it reached unity, and finally increased again (Figure 3A). The saccadic gain increased faster during the second gain-up session than during the first(Figure 3B). Therefore, the reversal learning did not washout the system. The Bayesian learner shows a similar phenomenon and provides a rationale: At the end of the first gain-up session for the Bayesian learner, most of the gain change is associated with a slow timescale (Figure 3C). In the subsequent gain-down session, errors produce rapid changes in the fast timescales so that by the time the gain estimate reaches unity, the fast and slow timescales have opposite estimates. Therefore, the gain-down session did not reset the system, but the latent variables store the history of adaptation. In the subsequent gain-up session, the rate of re-adaptation is faster than initial adaptation because the fast timescales decay upwards in between trials (Figure 3D). After about 100 saccades the speed gain from the low frequencies is over and is turned into a slowed increase due to the decreased error term.

In a second experiment, Kojima et al [12] found that saccade gains could change despite the fact that the animal was provided with no feedback to guide its performance. In this experiment the monkeys were again trained in a gain-up following by a gain-down session. Afterwards they spent some time in the dark. When they come out of the dark their gain had spontaneously increased (Figure 3E). The same effect is seen for the Bayesian learner (Figure 3F). In the dark period, the system makes no observations and therefore cannot learn from error. However, the estimates are still affected by their timescales of change: the estimate moves up fast along the fast timescales but slowly along the slow timescales. At the start of the darkness period there is a positive upward and a negative downward disturbance inferred by the system (Figure 1C, reversal). Consequently, by the end of the dark period, the estimate has become gain-up, the gain learned in the initial session. This produces the apparent spontaneous recovery observed in Figure 3F. Updating without feedback leads the system to infer unobserved dynamics of the oculomotor plant and these dynamics lead to the observed changes.

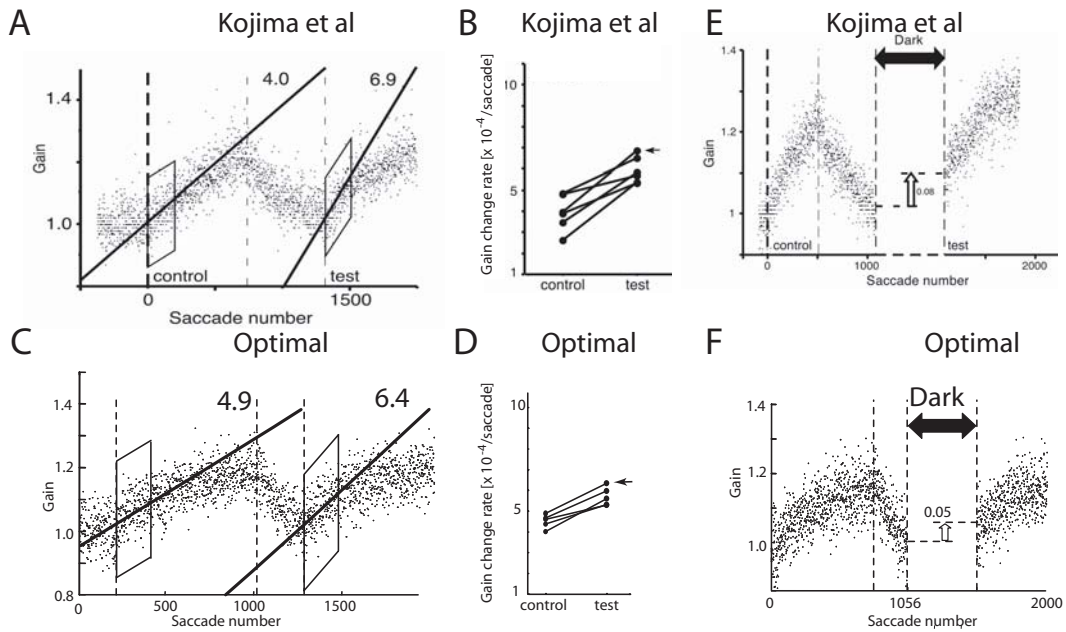

Figure 3: The double reversal paradigm. A) The gain is first adapted up until it reaches about 1.2 with a negative target jump of 35%. Then it is adapted down with a positive target jump of 35%. Once the gain reaches 1 again it is adapted up with a positive target jump again. Data reprinted from [12] with permission. B) The speed of adaptation is compared between the first adaptation and the second positive adaptation. C) the same as in A) for the Bayesian learner. D) the same as in B for the Bayesian learner. E) Double reversal paradigm with darkness, reprinted from [12]. The gain used by the monkey is changing during this interval. F) The same graph is shown for the Bayesian learner.

# 3 Discussion

Traditional models of adaptation simply change motor commands to reduce prediction errors [13]. Our approach differs from traditional approaches in three major ways. (1) The system represents its knowledge of the properties of the motor system at different timescales and explicitly models how these disturbances evolve over time. (2) It represents the uncertainty it has about the magnitude of the disturbances. (3) It formulates the computational aim of adaptation in terms of optimally predicting ongoing changes in the properties of the motor plant.

Multiple studies address each single of these points on its own. Multi-timescale learning is a classical phenomenon described frequently [14, 8]. Two timescales had been proposed in the context of connectionist learning theory [11]. In the context of motor adaptation Smith et al. [10] proposed a model where the motor system responds to error with two systems: one that is highly sensitive to error but rapidly forgets and another that has poor sensitivity to error but has strong retention. In the context of classical conditioning, it has been proposed that the nervous system should keep a measure of uncertainty about its current parameter estimates to allow an optimal combination of new information with current knowledge [15]. Even the earliest studies of oculomotor adaptation realized that the objective of adaptation is to allow precise movement with a relentlessly changing motor plant [3]. Our approach unifies these ideas in a consistent computational framework and explains a wide range of experiments.

Multi timescale adaptation and learning is a near universal phenomenon [14, 8, 16, 17]. Within the area of psychology it was found that learning follows multiscale behavior [17]. It has been proposed that multiscale learning may arise from chunking effects [14, 18]. The work presented here suggests a different interpretation. Multiscale learning in cognitive systems may be a result of a system that has originally evolved to deal with ever changing motor problems. Multiscale adaptation can also be seen in the way visual neurons adapt to changing visual stimuli [16]. The phenomenon of spontaneous recovery in classical conditioning [19, 20] is largely equivalent to the findings of Kojima et al [12] and can also be explained within the Bayesian multiscale learner framework.

The presented model obviously does not explain all known effects in motor or even saccadic gain adaptation. For example it has been found that adapting up usually has a somewhat different time-course to adapting down [21, 16, 12]. Moreover it seems that adaptation speed of monkeys can be very different on one day than the other and from one experimental setting to the other (e.g. Figure 2E and F). In learning reach control, there is more direct evidence that people can actually modify their rates of adaptation as a function of the auto-correlations of the perturbation [22]. This can be seen as the system learning about the size of the change parameter $\sigma_\tau$ in this theory. Moreover, we certainly estimate the uncertainty we have about a visual stimulus in a continuous fashion: uncertainty is smallest for a high contrast stimulus in our fovea and progressively larger with decreasing contrast and increasing eccentricity.

An important question for further enquiry is how the nervous system solves problems that require multiple timescale adaptation. The necessary effects could potentially be implemented directly by synapses that could exhibit LTP with powerlaw characteristics [23, 24]. Alternatively, small groups of neurons may jointly represent the estimates along with their uncertainties.

In summary, if we begin with the assumption that the nervous system optimally solves the problem of producing reliable movements with a motor plant that is affected by perturbations that have multiple timescales, then the learner will exhibit numerous properties that appear to match those reported in saccade and reach adaptation experiments.

# References

[1] W Becker. Metrics. In R. H. Wurtz and M Goldberg, editors, *The Neurobiology of Saccadic Eye Movements*, pages 13–67. Elsevier, Amsterdam, 1989.

[2] J. J. Hopp and A. F. Fuchs. The characteristics and neuronal substrate of saccadic eye movement plasticity. *Prog Neurobiol*, 72(1):27–53, 2004.

[3] SC McLaughlin. Parametric adjustment in saccadic eye movement. *Percept. Psychophys.*, 2:359–362, 1967.

[4] J. Wallman and A. F. Fuchs. Saccadic gain modification: visual error drives motor adaptation. *J Neurophysiol*, 80(5):2405–16, 1998.

[5] D. O. Bahcall and E. Kowler. Illusory shifts in visual direction accompany adaptation of saccadic eye movements. *Nature*, 400(6747):864–6, 1999.

[6] R. E. Kalman. A new approach to linear filtering and prediction problems. *J. of Basic Engineering (ASME)*, 82D:35–45, 1960.

[7] F. R. Robinson, R. Soetedjo, and C. Noto. Distinct short-term and long-term adaptation to reduce saccade size in monkey. *J Neurophysiol*, 2006.

[8] K. M. Newell. Motor skill acquisition. *Annu Rev Psychol*, 42:213–37, 1991.

[9] J. W. Krakauer, C. Ghez, and M. F. Ghilardi. Adaptation to visuomotor transformations: consolidation, interference, and forgetting. *J Neurosci*, 25(2):473–8, 2005.

[10] A.M. Smith, A. Ghazzizadeh, and R. Shadmehr. Interacting adaptive processes with different timescales underlie short-term motor learning. *PLoS Biol*, 4(e179), 2006.

[11] G. Hinton and C. Plaut. Using fast weights to deblur old memories. In Erlbaum, editor, *9th Annual Conference of the Cognitive Science Society*, pages 177–186, Hillsdale,NJ, 1987.

[12] Y. Kojima, Y. Iwamoto, and K. Yoshida. Memory of learning facilitates saccadic adaptation in the monkey. *J Neurosci*, 24(34):7531–9, 2004.

[13] K. A. Thoroughman and R. Shadmehr. Learning of action through adaptive combination of motor primitives. *Nature*, 407(6805):742–7, 2000.

[14] John R. Anderson. *The adaptive character of thought*. Erlbaum, Hillsdale, NJ, 1990.

[15] A. J. Yu and P. Dayan. Uncertainty, neuromodulation, and attention. *Neuron*, 46(4):681–92, 2005.

[16] A. L. Fairhall, G. D. Lewen, W. Bialek, and R. R. de Ruyter Van Steveninck. Efficiency and ambiguity in an adaptive neural code. *Nature*, 412(6849):787–92, 2001.

[17] H.P. Bahrick, L.E. Bahrick, A.S. Bahrick, and P.O. Bahrick. Maintenance of foreign language vocabulary and the spacing effect. *Psychological Science*, 4:31321, 1993.

[18] P. I. Pavlik and J. R. Anderson. An act-r model of the spacing effect. In F. Detje, D. Doerner, and H. Schaub, editors, *In Proceedings of the Fifth International Conference on Cognitive Modeling*, pages 177–182, Bamberg, Germany, 2003. Universitats-Verlag Bamberg.

[19] D. C. Brooks and M. E. Bouton. A retrieval cue for extinction attenuates spontaneous recovery. *J Exp Psychol Anim Behav Process*, 19(1):77–89, 1993.

[20] R. A. Rescorla. Spontaneous recovery varies inversely with the training-extinction interval. *Learn Behav*, 32(4):401–8, 2004.

[21] J. M. Miller, T. Anstis, and W. B. Templeton. Saccadic plasticity: parametric adaptive control by retinal feedback. *J Exp Psychol Hum Percept Perform*, 7(2):356–66, 1981.

[22] M. Smith, E. Hwang, and R. Shadmehr. Learning to learn- optimal adjustment of the rate at which the motor system adapts. In *In Proceedings of the Society for Neuroscience*, 2004.

[23] C. A. Barnes. Memory deficits associated with senescence: a neurophysiological and behavioral study in the rat. *J Comp Physiol Psychol*, 93(1):74–104, 1979.

[24] S. Fusi, P. J. Drew, and L. F. Abbott. Cascade models of synaptically stored memories. *Neuron*, 45(4):599–611, 2005.
